# How to Describe Neuronal Activity: Spikes, Rates, or Assemblies?

**Wulfram Gerstner and J. Leo van Hemmen**
Physik-Department der TU München
D-85748 Garching bei München, Germany

## Abstract

What is the 'correct' theoretical description of neuronal activity? The analysis of the dynamics of a globally connected network of spiking neurons (the Spike Response Model) shows that a description by mean firing *rates* is possible only if active neurons fire incoherently. If firing occurs coherently or with spatio-temporal correlations, the *spike* structure of the neural code becomes relevant. Alternatively, neurons can be gathered into local or distributed ensembles or '*assemblies*'. A description based on the mean ensemble activity is, in principle, possible but the interaction between different assemblies becomes highly nonlinear. A description with spikes should therefore be preferred.

## 1 INTRODUCTION

Neurons communicate by sequences of short pulses, the so-called action potentials or spikes. One of the most important problems in theoretical neuroscience concerns the question of how information on the environment is encoded in such spike trains: Is the exact timing of spikes with relation to earlier spikes relevant (*spike* or *interval code* (MacKay and McCulloch 1952) or does the mean firing rate averaged over several spikes contain all important information (*rate code*; see, e.g., Stein 1967)? Are spikes of single neurons important or do we have to consider ensembles of equivalent neurons (*ensemble code*)? If so, can we find local ensembles (e.g., columns; Hubel and Wiesel 1962) or do neurons form 'assemblies' (Hebb 1949) distributed all over the network?

## 2   SPIKE RESPONSE MODEL

We consider a globally connected network of $N$ neurons with $1 \leq i \leq N$. A neuron $i$ fires, if its membrane potential passes a threshold $\theta$. A spike at time $t_i^f$ is described by a $\delta$-pulse; thus $S_i^F(t) = \sum_{f=1}^{F} \delta(t - t_i^f)$ is the spike train of neuron $i$. Spikes are labelled such that $t_i^1$ is the most recent spike and $t_i^F$ is the $F^{th}$ spike going back in time.

In the Spike Response Model, short SRM, (Gerstner 1990, Gerstner and van Hemmen 1992) a neuron is characterized by two different *response functions*, $\epsilon$ and $\eta^{ref}$. Spikes which neuron $i$ receives from other neurons evoke a synaptic potential

$$h_i^{syn}(t) = \sum_{j=1}^{N} J_{ij} \int_0^{\infty} \epsilon(s) S_j^F(t - s) ds \qquad (1)$$

where the response kernel

$$\epsilon(s) = \begin{cases} 0 & \text{for } s \leq \Delta^{tr} \\ \frac{s - \Delta^{tr}}{\tau_s^2} \exp\left(-\frac{s - \Delta^{tr}}{\tau_s}\right) & \text{for } s > \Delta^{tr} \end{cases} \qquad (2)$$

describes a typical excitatory or inhibitory postsynaptic potential; see Fig. 1. The weight $J_{ij}$ is the synaptic efficacy of a connection from $j$ to $i$, $\Delta^{tr}$ is the axonal (and synaptic) transmission time, and $\tau_s$ is a time constant of the postsynaptic neuron. The origin $s = 0$ in (2) denotes the firing time of a presynaptic spike. In simulations we usually assume $\tau_s = 2\ ms$ and for $\Delta^{tr}$ a value between 1 and 4 $ms$

Similarly, spike emission induces refractoriness immediately after spiking. This is modelled by a refractory potential

$$h_i^{ref}(t) = \int_0^{\infty} \eta^{ref}(s) S_i^1(t - s) ds \qquad (3)$$

with a refractory function

$$\eta^{ref}(s) = \begin{cases} -\infty & \text{for } s \leq \gamma^{ref} \\ \eta_0/(s - \gamma^{ref}) & \text{for } s > \gamma^{ref}. \end{cases} \qquad (4)$$

For $0 \leq s \leq \gamma^{ref}$ the neuron is in the absolute refractory period and cannot spike at all whereas for $s > \gamma^{ref}$ spiking is possible but difficult (relative refractory period). To put it differently, $\theta - \eta^{ref}(s)$ describes an increased threshold immediately after spiking; cf. Fig. 1. In simulations, $\gamma^{ref}$ is taken to be 4 $ms$. Note that, for the sake of simplicity, we assume that only the most recent spike $S_i^1$ induces refractoriness whereas all past spikes $S_i^F$ contribute to the synaptic potential; cf., Eqs. (1) and (3).

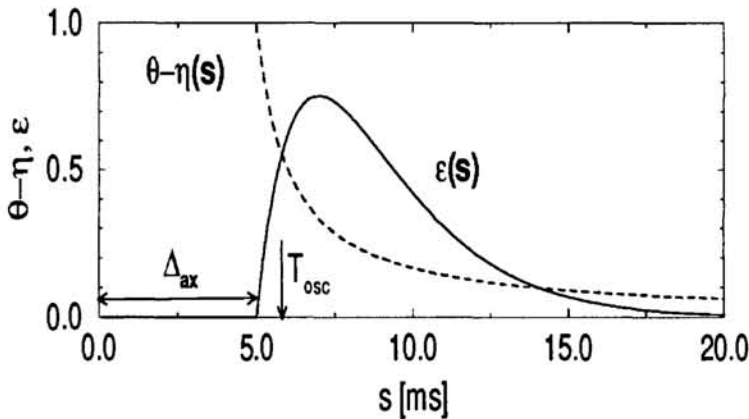

**Fig 1** *Response functions.* Immediately after firing at $s = 0$ the effective threshold is increased to $\theta - \eta^{ref}(s)$ (dashed). The form of an excitatory post-synaptic potential (EPSP) is described by the response function $\epsilon(s)$ (solid). It is delayed by a time $\Delta^{tr}$. The arrow denotes the period $T_{osc}$ of coherent oscillations; cf. Section 5.

The total membrane potential is the sum of both parts, i.e.

$$h_i(t) = h_i^{ref}(t) + h_i^{syn}(t). \tag{5}$$

Noise is included by introduction of a firing probability

$$P_F(h; \delta t) = \tau^{-1}(h)\,\delta t. \tag{6}$$

where $\delta t$ is an infinitesimal time interval and $\tau(h)$ is a time constant which depends on the momentary value of the membrane potential in relation to the threshold $\theta$. In analogy to the chemical reaction constant we assume

$$\tau(h) = \tau_0 \exp[-\beta(h - \theta)], \tag{7}$$

where $\tau_0$ is the response time at threshold. The parameter $\beta$ determines the amount of noise in the system. For $\beta \to \infty$ we recover the noise-free behavior, i.e., a neuron fires immediately, if $h > \theta$ ($\tau \to 0$), but it cannot fire, if $h < \theta$ ($\tau \to \infty$). Eqs. (1), (3), (5), and (6) define the spiking dynamics in a network of SRM-neurons.

## 3   FIRING STATISTICS

We start our considerations with a large ensemble of *identical* neurons driven by the same arbitrary synaptic potential $h^{syn}(t)$. We assume that all neurons have fired a first spike at $t = t_1^f$. Thus the total membrane potential is $h(t) = h^{syn}(t) + \eta^{ref}(t - t_1^f)$. If $h(t)$ slowly approaches $\theta$, some of the neurons will fire again. We now ask for the probability that a neuron which has fired at time $t_1^f$ will fire again at a later time $t$. The conditional probability $P_F^{(2)}(t|t_1^f)$ that the *next* spike of a given neuron occurs at time $t > t_1^f$ is

$$P_F^{(2)}(t|t_1^f) = \tau^{-1}[h(t)] \exp\left\{-\int_{t_1^f}^{t} \tau^{-1}[h(s')]ds'\right\}. \tag{8}$$

The exponential factor is the portion of neurons that have survived from time $t_1^f$ to time $t$ without firing again and the prefactor $\tau^{-1}[h(t)]$ is the instantaneous firing probability (6) at time $t$. Since the refractory potential is reset after each spike, the spiking statistics does not depend on earlier spikes, in other words, it is fully described by $P_F^{(2)}(t|t_1^f)$. This will be used below; cf. Eq. (14).

As a special case, we may consider constant synaptic input $h^{syn} \equiv h^0$. In this case, (8) yields the distribution of inter-spike intervals in a spike train of a neuron driven by constant input $h^0$. The mean firing rate at an input level $h^0$ is defined as the inverse of the mean inter-spike interval. Integration by parts yields

$$f[h^0] = \left\{ \int_{t_1^f}^{\infty} dt (t - t_1^f) P_F^{(2)}(t|t_1^f) \right\}^{-1} = \left\{ \int_0^{\infty} ds \exp\{ -\int_0^s \tau^{-1}[h^0 + \eta^{ref}(s')]ds' \} \right\}^{-1}. \tag{9}$$

Thus both firing rate and interval distribution can be calculated for arbitrary inputs.

# 4  ASSEMBLY FORMATION AND NETWORK DYNAMICS

We now turn to a large, but structured network. Structure is induced by the formation of different assemblies in the system. Each neuronal assembly $\alpha^\mu$ (Hebb 1949) consists of neurons which have the tendency to be active at the same time. Following the traditional interpretation, active means an elevated mean firing rate during some reasonable period of time. Later, in Section 5.3, we will deal with a different interpretation where active means a spike within a time window of a few *ms*. In any case, the notion of simultaneous activity allows to define an activity pattern $\{\xi_i^\mu, 1 \le i \le N\}$ with $\xi_i^\mu = 1$ if $i \in \alpha^\mu$ and $\xi_i^\mu = 0$ otherwise. Each neuron may belong to different assemblies $1 \le \mu \le q$. The vector $\boldsymbol{\xi}_i = (\xi_i^1, \ldots, \xi_i^q)$ is the 'identity card' of neuron $i$, e.g., $\boldsymbol{\xi}_i = (1,0,0,1,0)$ says that neuron $i$ belongs to assembly 1 and 4 but not to assembly 2,3, and 5.

Note that, in general, there are many neurons with the same identity card. This can be used to define ensembles (or sublattices) $L(\mathbf{x})$ of equivalent neurons, i.e., $L(\mathbf{x}) = \{i | \boldsymbol{\xi}_i = \mathbf{x}\}$ (van Hemmen and Kühn 1991). In general, the number of neurons $|L(\mathbf{x})|$ in an ensemble $L(\mathbf{x})$ goes to infinity if $N \to \infty$, and we write $|L(\mathbf{x})| = p(\mathbf{x})N$. The mean *activity* of an ensemble $L(\mathbf{x})$ can be defined by

$$A(\mathbf{x}, t) = \lim_{\Delta t \to 0} \lim_{N \to \infty} |L(\mathbf{x})|^{-1} \sum_{i \in L(\mathbf{x})} \int_t^{t + \Delta t} S_i^F(t)dt. \tag{10}$$

In the following we assume that the synaptic efficacies have been adjusted according to some Hebbian learning rule in a way that allows to stabilize the different activity patterns or assemblies $\alpha^\mu$. To be specific, we assume

$$J_{ij} = \frac{J_0}{N} \sum_{\mu=1}^{q} \sum_{\nu=1}^{q} Q_{\mu\nu} post(\xi_i^\mu) pre(\xi_j^\nu) \tag{11}$$

where $post(x)$ and $pre(x)$ are some arbitrary functions characterizing the pre- and postsynaptic part of synaptic learning. Note that for $Q_{\mu\nu} = \delta_{\mu\nu}$ and $post(x)$ and $pre(x)$ linear, Eq. (11) can be reduced to the usual Hebb rule.

With the above definitions we can write the synaptic potential of a neuron $i \in L(\mathbf{x})$ in the following form

$$h^{syn}(\mathbf{x}, t) = J_0 \sum_{\mu=1}^{q} \sum_{\nu=1}^{q} Q_{\mu\nu} post(x^\mu) \sum_{\mathbf{z}} pre(z^\nu) \int_0^{\infty} \epsilon(s') p(\mathbf{z}) A(\mathbf{z}, t - s')ds'. \tag{12}$$

We note that the index $i$ and $j$ has disappeared and there remains a dependence upon $\mathbf{x}$ and $\mathbf{z}$ *only*. The activity of a typical ensemble is given by (Gerstner and van Hemmen 1993, 1994)

$$A(\mathbf{x},t) = \int_0^\infty P_F^{(2)}(t|t-s)A(\mathbf{x},t-s)ds \qquad (13)$$

where

$$P_F^{(2)}(t|t-s) = \tau^{-1}[h^{syn}(\mathbf{x},t)+\eta^{ref}(s)] \exp\left\{ -\int_0^s \tau^{-1}[h^{syn}(\mathbf{x},t-s+s')+\eta^{ref}(s')]ds' \right\} \qquad (14)$$

is the conditional probability (8) that a neuron $i \in L(\mathbf{x})$ which has fired at time $t-s$ fires again at time $t$. Equations (12) - (14) define the ensemble dynamics of the network.

## 5   DISCUSSION

### 5.1   ENSEMBLE CODE

Equations. (12) - (14) show that in a large network a description by mean ensemble activities is, in principle, possible. A couple of things, however, should be noted. First, the interaction between the activity of different ensembles is highly nonlinear. It involves three integrations over the past and one exponentiation; cf. (12) - (14). If we had *started* theoretical modeling with an approach based on mean activities, it would have been hard to find the correct interaction term.

Second, $L(\mathbf{x})$ defines an ensemble of equivalent neurons which is a *subset* of a given assembly $\alpha^\mu$. A reduction of (12) to pure assembly activities is, in general, not possible. Finally, equivalent neurons that form an ensemble $L(\mathbf{x})$ are not necessarily situated next to each other. In fact, they may be distributed all over the network; cf. Fig. 2. In this case a *local* ensemble average yields meaningless results. A theoretical model based on local ensemble averaging is useful only if we know that neighboring neurons have the same 'identity card'.

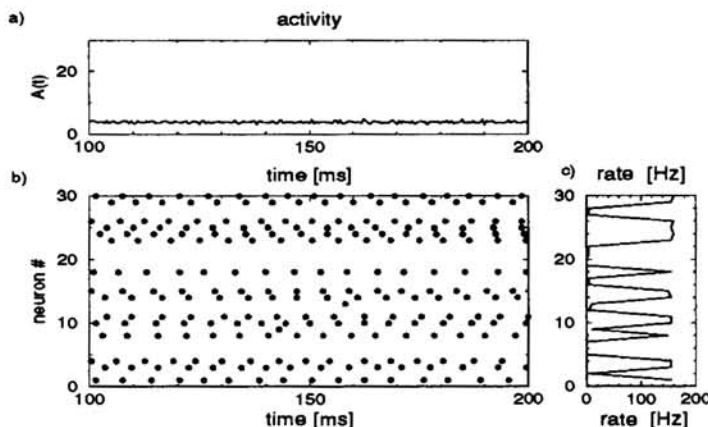

**Fig. 2**
Stationary activity (incoherent firing). In this case a description by firing rates is possible. (*a*) Ensemble averaged activity $A(\mathbf{x},t)$. (*b*) Spike raster of 30 neurons out of a network of 4000. (*c*) Time-averaged mean firing rate $f$. We have two different assemblies, one of them active ($\Delta^{tr} = 2\ ms$, $\beta = 5$).

### 5.2   RATE CODE

Can the system of Eqs. (12) -(14) be transformed into a rate description? In general, this is not the case but if we *assume* that the ensemble activities are constant in

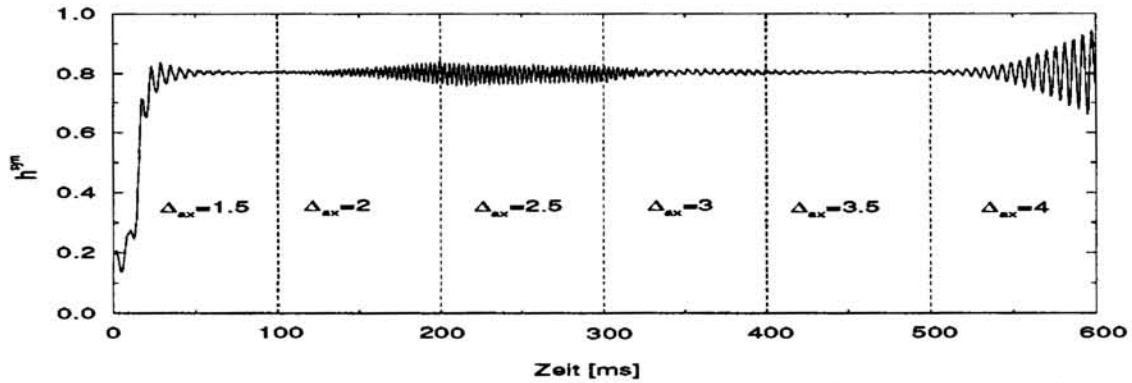

**Fig. 3** *Stability of stationary states.*The postsynaptic potential $h^{syn}$ is plotted as a function of time. Every $100\ ms$ the delay $\Delta^{tr}$ has been increased by $0.5\ ms$. In the stationary state ($\Delta^{tr} = 1.5\ ms$ and $\Delta^{tr} = 3.5\ ms$), active neurons fire regularly with rate $T_p^{-1} = 1/5.5\ ms$. For a delay $\Delta^{tr} > 3.5\ ms$, oscillations with period $\omega_1 = 2\pi/T_p$ build up rapidly. For intermediate delays $2 \leq \Delta^{tr} \leq 2.5$ small-amplitude oscillations with twice the frequency occur. Higher harmonics are suppressed by noise ($\beta = 20$).

time, i.e., $A(\mathbf{x}, t) \equiv A(\mathbf{x})$, then an exact reduction is possible. The result is a fixed-point equation (Gerstner and van Hemmen 1992)

$$A(\mathbf{x}) = f[J_0 \sum_{\mu=1}^{q} \sum_{\nu=1}^{q} Q_{\mu\nu} post(x^\mu) \sum_{\mathbf{z}} pre(z^\nu)p(\mathbf{z})A(\mathbf{z})] \qquad (15)$$

where

$$f[h^{syn}] = \left\{ \int_0^\infty ds\, \exp\{-\int_0^s \tau^{-1}[h^{syn} + \eta^{ref}(s')]ds'\} \right\}^{-1} \qquad (16)$$

is the *mean firing rate* (9) of a typical neuron stimulated by a synaptic input $h^{syn}$. Constant activities correspond to *incoherent*, stationary firing and in this case a rate code is sufficient; cf. Fig. 2.

Two points should, however, be kept in mind. First, a stationary state of incoherent firing is not necessarily stable. In fact, in a noise-free system the stationary state is always *unstable* and oscillations build up (Gerstner and van Hemmen 1993). In a system with noise, the stability depends on the noise level $\beta$ and the delay $\Delta^{tr}$ of axonal and synaptic transmission (Gerstner and van Hemmen 1994). This is shown in Fig. 3 where the delay $\Delta^{tr}$ has been increased every $100\ ms$. The frequency of the small-amplitude oscillation around the stationary state is approximately equal to the mean firing rate (16) in the stationary state or higher harmonics thereof. A small-amplitude oscillation corresponds to partially synchronized activity. Note that for $\Delta^{tr} = 4\ ms$ a large-amplitude oscillation builds up. Here all neurons fire in nearly perfect synchrony; cf. Fig. 4. In the noiseless case $\beta \to \infty$, the oscillations period of such a collective or 'locked' oscillation can be found from the threshold condition

$$T_{osc} = \inf \left\{ s\,|\,\theta = \eta^{ref}(s) + J_0 \sum_{n=1}^{\infty} \epsilon(ns) \right\}. \qquad (17)$$

In most cases the contribution with $n = 1$ is dominant which allows a simple graphical solution. The first intersection of the effective threshold $\theta - \eta^{ref}(s)$ with the

weighted EPSP $J_0\epsilon(s)$ yields the oscillation period; cf. Fig 1. An analytical argument shows that locking is stable only if $\frac{d}{ds}\epsilon|_{T_{osc}} > 0$ (Gerstner and van Hemmen 1993).

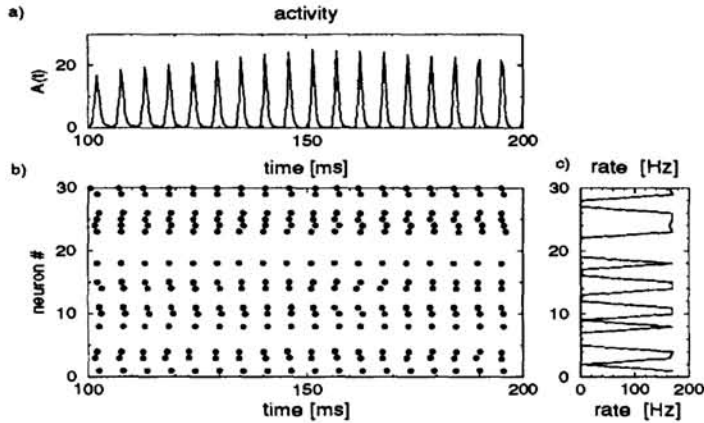

**Fig. 4**
Oscillatory activity (coherent firing). In this case a description by firing rates must be combined with a description by ensemble activities. ($a$) Ensemble averaged activity $A(\mathbf{x}, t)$. ($b$) Spike raster of 30 neurons out of a network of 4000. ($c$) Time-averaged mean firing rate $f$. In this simulation, we have used $\Delta^{tr} = 4 \ ms$ and $\beta = 8$.

Second, even if the incoherent state is stable and attractive, there is always a transition time before the stationary state is assumed. During this time, a rate description is insufficient and we have to go back to the full dynamic equations (12) - (14). Similarly, if neurons are subject to a fast time-dependent external stimulus, a rate code fails.

## 5.3 SPIKE CODE

A superficial inspection of Eqs. (12) - (14) gives the impression that all information about neuronal spiking has disappeared. This is, however, false. The term $A(\mathbf{x}, t-s)$ in (13) denotes all neurons with 'identity card' $\mathbf{x}$ that have fired at time $t-s$. The integration kernel in (13) is the conditional probability that one of these neurons fires again at time $t$. Keeping $t-s$ fixed and varying $t$ we get the distribution of inter-spike intervals for neurons in $L(\mathbf{x})$. Thus information on both *spikes* and *intervals* is contained in (13) and (14).

We can make use of this fact, if we consider network states where in every time step a *different* assembly is active. This leads to a spatio-temporal spike pattern as shown in Fig. 5. To transform a specific spike pattern into a stable state of the network we can use a Hebbian learning rule. However, in contrast to the standard rule, a synapse is strenthened only if pre- and postsynaptic activity occurs simultaneously within a time window of a few *ms* (Gerstner et al. 1993). Note that in this case, averaging over time or space spoils the information contained in the spike pattern.

## 5.4 CONCLUSIONS

Equations. (12) - (14) show that in our large and fully connected network an ensemble code with an *appropriately* chosen ensemble is sufficient. If, however, the efficacies (11) and the connection scheme become more involved, the construction of appropriate ensembles becomes more and more difficult. Also, in a finite network we cannot make use of the law of large number in defining the activities (10). Thus, in general, we should always start with a network model of *spiking* neurons.

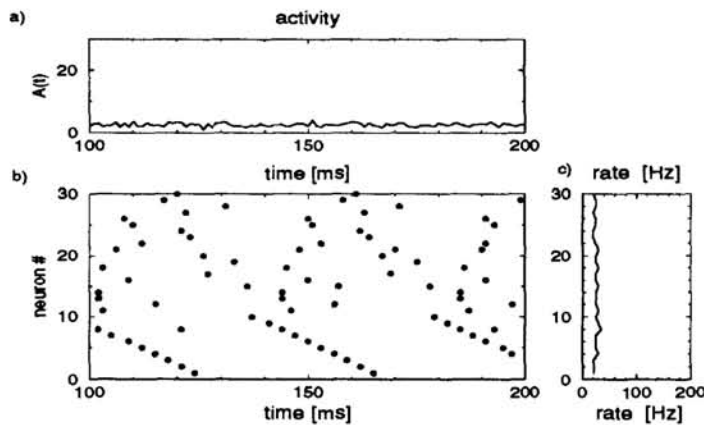

**Fig. 5**
Spatio-temporal spike pattern. In this case, neither firing rates nor locally averaged activities contain enough information to describe the state of the network. (*a*) Ensemble averaged activity $A(t)$. (*b*) Spike raster of 30 neurons out of a network of 4000. (*c*) Time-averaged mean firing rate $f$.

**Acknowledgements:** This work has been supported by the Deutsche Forschungs-gemeinschaft (DFG) under grant No. He 1729/2-1.

## References

Gerstner W (1990) Associative memory in a network of 'biological' neurons. In: *Advances in Neural Information Processing Systems 3*, edited by R.P. Lippmann, J.E. Moody, and D.S. Touretzky (Morgan Kaufmann, San Mateo, CA) pp 84-90

Gerstner W and van Hemmen JL (1992a) Associative memory in a network of 'spiking' neurons. Network 3:139-164

Gerstner W, van Hemmen JL (1993) Coherence and incoherence in a globally coupled ensemble of pulse-emitting units. Phys. Rev. Lett. **71**:312-315

Gerstner W, Ritz R, van Hemmen JL (1993b) Why spikes? Hebbian learning and retrieval of time-resolved excitation patterns. Biol. Cybern. **69**:503-515

Gerstner W and van Hemmen JL (1994) Coding and Information processing in neural systems. In: *Models of neural networks, Vol. 2*, edited by E. Domany, J.L. van Hemmen and K. Schulten (Springer-Verlag, Berlin, Heidelberg, New York) pp 1ff

Hebb DO (1949) *The Organization of Behavior.* Wiley, NewYork

van Hemmen JL and Kühn R(1991) Collective phenomena in neural networks. In: *Models of neural networks*, edited by E. Domany, J.L. van Hemmen and K. Schulten (Springer-Verlag, Berlin, Heidelberg, New York) pp 1ff

Hubel DH, Wiesel TN (1962) Receptive fields, binocular interaction and functional architecture in the cat's visual cortex. J. Neurophysiol. **28**:215-243

MacKay DM, McCulloch WS (1952) The limiting information capacity of a neuronal link. Bull. of Mathm. Biophysics **14**:127-135

Stein RB (1967) The information capacity of nerve cells using a frequency code. Biophys. J. **7**:797-826